# Multi-Bandit Best Arm Identification

**Victor Gabillon**  **Mohammad Ghavamzadeh**  **Alessandro Lazaric**
INRIA Lille - Nord Europe, Team SequeL
`{victor.gabillon,mohammad.ghavamzadeh,alessandro.lazaric}@inria.fr`

**Sébastien Bubeck**
Department of Operations Research and Financial Engineering, Princeton University
`sbubeck@princeton.edu`

## Abstract

We study the problem of identifying the best arm in each of the bandits in a multi-bandit multi-armed setting. We first propose an algorithm called Gap-based Exploration (GapE) that focuses on the arms whose mean is close to the mean of the best arm in the same bandit (i.e., small gap). We then introduce an algorithm, called GapE-V, which takes into account the variance of the arms in addition to their gap. We prove an upper-bound on the probability of error for both algorithms. Since GapE and GapE-V need to tune an exploration parameter that depends on the complexity of the problem, which is often unknown in advance, we also introduce variations of these algorithms that estimate this complexity online. Finally, we evaluate the performance of these algorithms and compare them to other allocation strategies on a number of synthetic problems.

## 1 Introduction

Consider a clinical problem with $M$ subpopulations, in which one should decide between $K_m$ options for treating subjects from each subpopulation $m$. A subpopulation may correspond to patients with a particular gene biomarker (or other risk categories) and the treatment options are the available treatments for a disease. The main objective here is to construct a rule, which recommends the best treatment for each of the subpopulations. These rules are usually constructed using data from clinical trials that are generally costly to run. Therefore, it is important to distribute the trial resources wisely so that the devised rule yields a good performance. Since it may take significantly more resources to find the best treatment for one subpopulation than for the others, the common strategy of enrolling patients as they arrive may not yield an overall good performance. Moreover, applying treatment options uniformly at random in a subpopulation could not only waste trial resources, but also it might run the risk of finding a bad treatment for that subpopulation. This problem can be formulated as the *best arm identification* over $M$ multi-armed bandits [1], which itself can be seen as the problem of *pure exploration* [4] over multiple bandits. In this formulation, each subpopulation is considered as a multi-armed bandit, each treatment as an arm, trying a medication on a patient as a pull, and we are asked to recommend an arm for each bandit after a given number of pulls (budget). The evaluation can be based on **1)** the average over the bandits of the reward of the recommended arms, or **2)** the average probability of error (not selecting the best arm), or **3)** the maximum probability of error. Note that this setting is different from the standard multi-armed bandit problem in which the goal is to maximize the cumulative sum of rewards (see e.g., [13, 3]).

The pure exploration problem is about designing strategies that make the best use of the limited budget (e.g., the total number of patients that can be admitted to the clinical trial) in order to optimize the performance in a decision-making task. Audibert et al. [1] proposed two algorithms to address this problem: **1)** a highly exploring strategy based on upper confidence bounds, called UCB-E, in which the optimal value of its parameter depends on some measure of the complexity of the problem, and **2)** a parameter-free method based on progressively rejecting the arms which seem to be suboptimal, called Successive Rejects. They showed that both algorithms are nearly optimal since their probability of returning the wrong arm decreases exponentially at a rate. Racing algorithms (e.g., [10, 12])

and action-elimination algorithms [7] address this problem under a constraint on the accuracy in identifying the best arm and they minimize the budget needed to achieve that accuracy. However, UCB-E and Successive Rejects are designed for a single bandit problem, and as we will discuss later, cannot be easily extended to the multi-bandit case studied in this paper. Deng et al. have recently proposed an active learning algorithm for resource allocation over multiple bandits [5]. However, they do not provide any theoretical analysis for their algorithm and only empirically evaluate its performance. Moreover, the target of their proposed algorithm is to minimize the maximum uncertainty in estimating the value of the arms for each bandit. Note that this is different than our target, which is to maximize the quality of the arms recommended for each bandit.

In this paper, we study the problem of best-arm identification in a multi-armed multi-bandit setting under a fixed budget constraint, and propose an algorithm, called Gap-based Exploration (GapE), to solve it. The allocation strategy implemented by GapE focuses on the gap of the arms, i.e., the difference between the mean of the arm and the mean of the best arm (in that bandit). The GapE-variance (GapE-V) algorithm extends this approach taking into account also the variance of the arms. For both algorithms, we prove an upper-bound on the probability of error that decreases exponentially with the budget. Since both GapE and GapE-V need to tune an exploration parameter that depends on the complexity of the problem, which is rarely known in advance, we also introduce their adaptive version. Finally, we evaluate the performance of these algorithms and compare them with *Uniform* and *Uniform+UCB-E* strategies on a number of synthetic problems. Our empirical results indicate that **1)** GapE and GapE-V have a better performance than *Uniform* and *Uniform+UCB-E*, and **2)** the adaptive version of these algorithms match the performance of their non-adaptive counterparts.

## 2 Problem Setup

In this section, we introduce the notation used throughout the paper and formalize the multi-bandit best arm identification problem. Let $M$ be the number of bandits and $K$ be the number of arms for each bandit (we use indices $m, p, q$ for the bandits and $k, i, j$ for the arms). Each arm $k$ of a bandit $m$ is characterized by a distribution $\nu_{mk}$ bounded in $[0, b]$ with mean $\mu_{mk}$ and variance $\sigma^2_{mk}$. In the following, we assume that each bandit has a unique best arm. We denote by $\mu^*_m$ and $k^*_m$ the mean and the index of the best arm of bandit $m$ (i.e., $\mu^*_m = \max_{1 \le k \le K} \mu_{mk}$, $k^*_m = \arg\max_{1 \le k \le K} \mu_{mk}$). In each bandit $m$, we define the gap for each arm as $\Delta_{mk} = |\max_{j \ne k} \mu_{mj} - \mu_{mk}|$.

The clinical trial problem described in Sec. 1 can be formalized as a game between a stochastic multi-bandit environment and a forecaster, where the distributions $\{\nu_{mk}\}$ are unknown to the forecaster. At each round $t = 1, \ldots, n$, the forecaster pulls a bandit-arm pair $I(t) = (m, k)$ and observes a sample drawn from the distribution $\nu_{I(t)}$ independent from the past. The forecaster estimates the expected value of each arm by computing the average of the samples observed over time. Let $T_{mk}(t)$ be the number of times that arm $k$ of bandit $m$ has been pulled by the end of round $t$, then the mean of this arm is estimated as $\widehat{\mu}_{mk}(t) = \frac{1}{T_{mk}(t)} \sum_{s=1}^{T_{mk}(t)} X_{mk}(s)$, where $X_{mk}(s)$ is the $s$-th sample observed from $\nu_{mk}$. Given the previous definitions, we define the estimated gaps as $\widehat{\Delta}_{mk}(t) = |\max_{j \ne k} \widehat{\mu}_{mj}(t) - \widehat{\mu}_{mk}(t)|$. At the end of round $n$, the forecaster returns for each bandit $m$ the arm with the highest estimated mean, i.e., $J_m(n) = \arg\max_k \widehat{\mu}_{mk}(n)$, and incurs a regret

$$r(n) = \frac{1}{M} \sum_{m=1}^{M} r_m(n) = \frac{1}{M} \sum_{m=1}^{M} \left( \mu^*_m - \mu_{mJ_m(n)} \right).$$

As discussed in the introduction, other performance measures can be defined for this problem. In some applications, returning the wrong arm is considered as an error independently from its regret, and thus, the objective is to minimize the average probability of error

$$e(n) = \frac{1}{M} \sum_{m=1}^{M} e_m(n) = \frac{1}{M} \sum_{m=1}^{M} \mathbb{P}\left( J_m(n) \ne k^*_m \right).$$

Finally, in problems similar to the clinical trial, a reasonable objective is to return the right treatment for all the genetic profiles and not just to have a small average probability of error. In this case, the global performance of the forecaster can be measured as

$$\ell(n) = \max_m \ell_m(n) = \max_m \mathbb{P}\left( J_m(n) \ne k^*_m \right).$$

It is interesting to note the relationship between these three performance measures: $\min_m \Delta_m \times e(n) \le \mathbb{E}r(n) \le b \times e(n) \le b \times \ell(n)$, where the expectation in the regret is w.r.t. the random samples. As a result, any algorithm minimizing the worst case probability of error, $\ell(n)$, also controls the average probability of error, $e(n)$, and the simple regret $\mathbb{E}r(n)$. Note that the algorithms introduced in this paper directly target the problem of minimizing $\ell(n)$.

> **Parameters:** number of rounds $n$, exploration parameter $a$, maximum range $b$
> **Initialize:** $T_{mk}(0) = 0$, $\widehat{\Delta}_{mk}(0) = 0$ for all bandit-arm pairs $(m,k)$
> **for** $t = 1, 2, \ldots, n$ **do**
>     Compute $B_{mk}(t) = -\widehat{\Delta}_{mk}(t-1) + b\sqrt{\frac{a}{T_{mk}(t-1)}}$ for all bandit-arm pairs $(m,k)$
>     Draw $I(t) \in \arg\max_{m,k} B_{mk}(t)$
>     Observe $X_{I(t)}\big(T_{I(t)}(t-1)+1\big) \sim \nu_{I(t)}$
>     Update $T_{I(t)}(t) = T_{I(t)}(t-1)+1$ and $\widehat{\Delta}_{mk}(t)$ $\forall k$ of the selected bandit
> **end for**
> Return $J_m(n) \in \arg\max_{k \in \{1,\ldots,K\}} \widehat{\mu}_{mk}(n)$, $\forall m \in \{1 \ldots M\}$

Figure 1: The pseudo-code of the gap-based Exploration (GapE) algorithm.

## 3 The Gap-based Exploration Algorithm

Fig. 1 contains the pseudo-code of the gap-based exploration (GapE) algorithm. GapE flattens the bandit-arm structure and reduces it to a single-bandit problem with $MK$ arms. At each time step $t$, the algorithm relies on the observations up to time $t-1$ to build an index $B_{mk}(t)$ for each bandit-arm pair, and then selects the pair $I(t)$ with the highest index. The index $B_{mk}$ consists of two terms. The first term is the negative of the estimated gap for arm $k$ in bandit $m$. Similar to other upper-confidence bound (UCB) methods [3], the second part is an exploration term which forces the algorithm to pull arms that have been less explored. As a result, the algorithm tends to pull arms with small estimated gap and small number of pulls. The exploration parameter $a$ tunes the level of exploration of the algorithm. As it is shown by the theoretical analysis of Sec. 3.1, if the time horizon $n$ is known, $a$ should be set to $a = \frac{4}{9}\frac{n-K}{H}$, where $H = \sum_{m,k} b^2/\Delta_{mk}^2$ is the *complexity* of the problem (see Sec. 3.1 for further discussion). Note that GapE differs from most standard bandit strategies in the sense that the $B$-index for an arm depends explicitly on the statistics of the other arms. This feature makes the analysis of this algorithm much more involved.

As we may notice from Fig. 1, GapE resembles the UCB-E algorithm [1] designed to solve the pure exploration problem in the single-bandit setting. Nonetheless, the use of the negative estimated gap $(-\widehat{\Delta}_{mk})$ instead of the estimated mean $(\widehat{\mu}_{mk})$ (used by UCB-E) is crucial in the multi-bandit setting. In the single-bandit problem, since the best and second best arms have the same gap $(\Delta_{mk_m^*} = \min_{k \neq k_m^*} \Delta_{mk})$, GapE considers them equivalent and tends to pull them the same amount of time, while UCB-E tends to pull the best arm more often than the second best one. Despite this difference, the performance of both algorithms in predicting the best arm after $n$ pulls would be the same. This is due to the fact that the probability of error depends on the capability of the algorithm to distinguish optimal and suboptimal arms, and this is not affected by a different allocation over the best and second best arms as long as the number of pulls allocated to that pair is large enough w.r.t. their gap. Despite this similarity, the two approaches become completely different in the multi-bandit case. In this case, if we run UCB-E on all the $MK$ arms, it tends to pull more the arm with the highest mean over all the bandits, i.e., $k^* = \arg\max_{m,k} \mu_{mk}$. As a result, it would be accurate in predicting the best arm $k^*$ over bandits, but may have an arbitrarily bad performance in predicting the best arm for each bandit, and thus, may incur a large error $\ell(n)$. On the other hand, GapE focuses on the arms with the smallest gaps. This way, it assigns more pulls to bandits whose optimal arms are difficult to identify (i.e., bandits with arms with small gaps), and as shown in the next section, it achieves a high probability in identifying the best arm in each bandit.

### 3.1 Theoretical Analysis

In this section, we derive an upper-bound on the probability of error $\ell(n)$ for the GapE algorithm.

**Theorem 1.** *If we run GapE with parameter* $0 < a \leq \frac{4}{9}\frac{n-MK}{H}$, *then its probability of error satisfies*

$$\ell(n) \leq \mathbb{P}\big(\exists m : J_m(n) \neq k_m^*\big) \leq 2MKn \exp(-\frac{a}{64}),$$

*in particular for* $a = \frac{4}{9}\frac{n-MK}{H}$, *we have* $\ell(n) \leq 2MKn \exp(-\frac{1}{144}\frac{n-MK}{H})$.

**Remark 1 (Analysis of the bound).** If the time horizon $n$ is known in advance, it would be possible to set the exploration parameter $a$ as a linear function of $n$, and as a result, the probability of error of GapE decreases exponentially with the time horizon. The other interesting aspect of the bound is the

complexity term $H$ appearing in the optimal value of the exploration parameter $a$ (i.e., $a = \frac{4}{9}\frac{n-K}{H}$). If we denote by $H_{mk} = b^2/\Delta_{mk}^2$, the complexity of arm $k$ in bandit $m$, it is clear from the definition of $H$ that each arm has an additive impact on the overall complexity of the multi-bandit problem. Moreover, if we define the complexity of each bandit $m$ as $H_m = \sum_k b^2/\Delta_{mk}^2$ (similar to the definition of complexity for UCB-E in [1]), the GapE complexity may be rewritten as $H = \sum_m H_m$. This means that the complexity of GapE is simply the sum of the complexities of all the bandits.

**Remark 2 (Comparison with the static allocation strategy).** The main objective of GapE is to tradeoff between allocating pulls according to the gaps (more precisely, according to the complexities $H_{mk}$) and the exploration needed to improve the accuracy of their estimates. If the gaps were known in advance, a nearly-optimal static allocation strategy assigns to each bandit-arm pair a number of pulls proportional to its complexity. Let us consider a strategy that pulls each arm a fixed number of times over the horizon $n$. The probability of error for this strategy may be bounded as

$$\ell_{\text{Static}}(n) \leq \mathbb{P}\big(\exists m : J_m(n) \neq k_m^*\big) \leq \sum_{m=1}^{M} \mathbb{P}\big(J_m(n) \neq k_m^*\big) \leq \sum_{m=1}^{M} \sum_{k \neq k_m^*} \mathbb{P}\big(\hat{\mu}_{mk_m^*}(n) \leq \hat{\mu}_{mk}(n)\big)$$

$$\leq \sum_{m=1}^{M} \sum_{k \neq k_m^*} \exp\big(-T_{mk}(n)\frac{\Delta_{mk}^2}{b^2}\big) = \sum_{m=1}^{M} \sum_{k \neq k_m^*} \exp\big(-T_{mk}(n)H_{mk}^{-1}\big). \quad (1)$$

Given the constraint $\sum_{mk} T_{mk}(n) = n$, the allocation minimizing the last term in Eq. 1 is $T_{mk}^*(n) = nH_{mk}/H$. We refer to this fixed strategy as *StaticGap*. Although this is not necessarily the optimal static strategy ($T_{mk}^*(n)$ minimizes an upper-bound), this allocation guarantees a probability of error smaller than $MK\exp(-n/H)$. Theorem 1 shows that, for $n$ large enough, GapE achieves the same performance as the static allocation *StaticGap*.

**Remark 3 (Comparison with other allocation strategies).** At the beginning of Sec. 3, we discussed the difference between GapE and UCB-E. Here we compare the bound reported in Theorem 1 with the performance of the *Uniform* and combined *Uniform+UCB-E* allocation strategies. In the uniform allocation strategy, the total budget $n$ is uniformly split over all the bandits and arms. As a result, each bandit-arm pair is pulled $T_{mk}(n) = n/(MK)$ times. Using the same derivation as in Remark 2, the probability of error $\ell(n)$ for this strategy may be bounded as

$$\ell_{\text{Unif}}(n) \leq \sum_{m=1}^{M} \sum_{k \neq k_m^*} \exp\big(-\frac{n}{MK}\frac{\Delta_{mk}^2}{b^2}\big) \leq MK\exp\big(-\frac{n}{MK\max_{m,k} H_{mk}}\big).$$

In the *Uniform+UCB-E* allocation strategy, i.e., a two-level algorithm that first selects a bandit uniformly and then pulls arms within each bandit using UCB-E, the total number of pulls for each bandit $m$ is $\sum_k T_{mk}(n) = n/M$, while the number of pulls $T_{mk}(n)$ over the arms in bandit $m$ is determined by UCB-E. Thus, the probability of error of this strategy may be bounded as

$$\ell_{\text{Unif+UCB-E}}(n) \leq \sum_{m=1}^{M} 2nK\exp\big(-\frac{n/M - K}{18H_m}\big) \leq 2nMK\exp\big(-\frac{n/M - K}{18\max_m H_m}\big),$$

where the first inequality follows from Theorem 1 in [1] (recall that $H_m = \sum_k b^2/\Delta_{mk}^2$). Let $b = 1$ (i.e., all the arms have distributions bounded in $[0, 1]$), up to constants and multiplicative factors in front of the exponentials, and if $n$ is large enough compared to $M$ and $K$ (so as to approximate $n/M - K$ and $n - K$ by $n$), the probability of error for the three algorithms may be bounded as

$$\ell_{\text{Unif}}(n) \leq \exp\Big(O\big(\frac{-n/MK}{\max\limits_{m,k} H_{mk}}\big)\Big), \quad \ell_{\text{U+UCBE}}(n) \leq \exp\Big(O\big(\frac{-n/M}{\max\limits_{m} H_m}\big)\Big), \quad \ell_{\text{GapE}}(n) \leq \exp\Big(O\big(\frac{-n}{\sum\limits_{m,k} H_{mk}}\big)\Big).$$

By comparing the arguments of the exponential terms, we have the trivial sequence of inequalities $MK\max_{m,k} H_{mk} \geq M\max_m \sum_k H_{mk} \geq \sum_{m,k} H_{mk}$, which implies that the upper bound on the probability of error of GapE is usually significantly smaller. This relationship, which is confirmed by the experiments reported in Sec. 4, shows that GapE is able to adapt to the complexity $H$ of the overall multi-bandit problem better than the other two allocation strategies. In fact, while the performance of the *Uniform* strategy depends on the most *complex* arm over the bandits and the strategy *Unif+UCB-E* is affected by the most complex bandit, the performance of GapE depends on the sum of the complexities of all the arms involved in the pure exploration problem.

*Proof of Theorem 1.* **Step 1.** Let us consider the following event:

$$\mathcal{E} = \left\{ \forall m \in \{1, \dots, M\}, \ \forall k \in \{1, \dots, K\}, \ \forall t \in \{1, \dots, n\}, \ |\widehat{\mu}_{mk}(t) - \mu_{mk}| < bc\sqrt{\frac{a}{T_{mk}(t)}} \right\}.$$

From Chernoff-Hoeffding's inequality and a union bound, we have $\mathbb{P}(\xi) \geq 1 - 2MKn \exp(-2ac^2)$. Now we would like to prove that on the event $\mathcal{E}$, we find the best arm for all the bandits, i.e., $J_m(n) = k_m^*$, $\forall m \in \{1 \dots M\}$. Since $J_m(n)$ is the empirical best arm of bandit $m$, we should prove that for any $k \in \{1, \dots, K\}$, $\widehat{\mu}_{mk}(n) \leq \widehat{\mu}_{mk_m^*}(n)$. By upper-bounding the LHS and lower-bounding the RHS of this inequality, we note that it would be enough to prove $bc\sqrt{a/T_{mk}(n)} \leq \Delta_{mk}/2$ on the event $\mathcal{E}$, or equivalently, to prove that for any bandit-arm pair $m, k$, we have $T_{mk}(n) \geq \frac{4ab^2c^2}{\Delta_{mk}^2}$.

**Step 2.** In this step, we show that in GapE, for any bandits $(m, q)$ and arms $(k, j)$, and for any $t \geq MK$, the following dependence between the number of pulls of the arms holds

$$-\Delta_{mk} + (1 + d)b\sqrt{\frac{a}{\max\left(T_{mk}(t) - 1, 1\right)}} \geq -\Delta_{qj} + (1 - d)b\sqrt{\frac{a}{T_{qj}(t)}}, \qquad (2)$$

where $d \in [0, 1]$. We prove this inequality by induction.

*Base step.* We know that after the first $MK$ rounds of the GapE algorithm, all the arms have been pulled once, i.e., $T_{mk}(t) = 1$, $\forall m, k$, thus if $a \geq 1/4d^2$, the inequality (2) holds for $t = MK$.

*Inductive step.* Let us assume that (2) holds at time $t - 1$ and we pull arm $i$ of bandit $p$ at time $t$, i.e., $I(t) = (p, i)$. So at time $t$, the inequality (2) trivially holds for every choice of $m$, $q$, $k$, and $j$, except when $(m, k) = (p, i)$. As a result, in the inductive step, we only need to prove that the following holds for any $q \in \{1, \dots M\}$ and $j \in \{1, \dots K\}$

$$-\Delta_{pi} + (1 + d)b\sqrt{\frac{a}{\max\left(T_{pi}(t) - 1, 1\right)}} \geq -\Delta_{qj} + (1 - d)b\sqrt{\frac{a}{T_{qj}(t)}}. \qquad (3)$$

Since arm $i$ of bandit $p$ has been pulled at time $t$, we have that for any bandit-arm pair $(q, j)$

$$-\widehat{\Delta}_{pi}(t - 1) + b\sqrt{\frac{a}{T_{pi}(t - 1)}} \geq -\widehat{\Delta}_{qj}(t - 1) + b\sqrt{\frac{a}{T_{qj}(t - 1)}}. \qquad (4)$$

To prove (3), we first prove an upper-bound for $-\widehat{\Delta}_{pi}(t - 1)$ and a lower-bound for $-\widehat{\Delta}_{qj}(t - 1)$

$$-\widehat{\Delta}_{pi}(t-1) \leq -\Delta_{pi} + \frac{2bc}{1 - c}\sqrt{\frac{a}{T_{pi}(t) - 1}} \quad \text{and} \quad -\widehat{\Delta}_{qj}(t-1) \geq -\Delta_{qj} - \frac{2\sqrt{2}bc}{1 - d}\sqrt{\frac{a}{T_{qj}(t)}}. \quad (5)$$

We report the proofs of the inequalities in (5) in App. B of [8]. The inequality (3), and as a result, the inductive step is proved by replacing $-\widehat{\Delta}_{pi}(t-1)$ and $-\widehat{\Delta}_{qj}(t-1)$ in (4) from (5) and under the conditions that $d \geq \frac{2c}{1-c}$ and $d \geq \frac{2\sqrt{2}c}{1-d}$. These conditions are satisfied by $d = 1/2$ and $c = \sqrt{2}/16$.

**Step 3.** In order to prove the condition of $T_{mk}(n)$ in step 1, we need to find a lower-bound on the number of pulls of all the arms at time $t = n$ (at the end). Let us assume that arm $k$ of bandit $m$ has been pulled less than $\frac{ab^2(1-d)^2}{\Delta_{mk}^2}$, which indicates that $-\Delta_{mk} + (1 - d)b\sqrt{\frac{a}{T_{mk}(n)}} > 0$. From this result and (2), we have $-\Delta_{qj} + (1 + d)b\sqrt{\frac{a}{T_{qj}(n)-1}} > 0$, or equivalently $T_{qj}(n) < \frac{ab^2(1+d)^2}{\Delta_{qj}^2} + 1$ for any pair $(q, j)$. We also know that $\sum_{q,j} T_{qj}(n) = n$. From these, we deduce that $n - MK < ab^2(1+d)^2 \sum_{q,j} \frac{1}{\Delta_{qj}^2}$. So, if we select $a$ such that $n - MK \geq ab^2(1+d)^2 \sum_{q,j} \frac{1}{\Delta_{qj}^2}$, we contradict the first assumption that $T_{mk}(n) < \frac{ab^2(1-d)^2}{\Delta_{mk}^2}$, which means that $T_{mk}(n) \geq \frac{4ab^2c^2}{\Delta_{mk}^2}$ for any pair $(m, k)$, when $1 - d \geq 2c$. This concludes the proof. The condition for $a$ in the statement of the theorem comes from our choice of $a$ in this step and the values of $c$ and $d$ from the inductive step. $\square$

### 3.2 Extensions

In this section we propose two variants on the GapE algorithm with the objective of extending its applicability and improving its performance.

**GapE with variance (GapE-V).** The allocation strategy implemented by GapE focuses only on the arms with small gap and does not take into consideration their variance. However, it is clear that the arms with small variance, even if their gap is small, just need a few pulls to be correctly estimated. In order to take into account both the gaps and variances of the arms, we introduce the GapE-variance (GapE-V) algorithm. Let $\widehat{\sigma}^2_{mk}(t) = \frac{1}{T_{mk}(t)-1} \sum_{s=1}^{T_{mk}(t)} X^2_{mk}(s) - \widehat{\mu}^2_{mk}(t)$ be the estimated variance for arm $k$ of bandit $m$ at the end of round $t$. GapE-V uses the following B-index for each arm:

$$B_{mk}(t) = -\widehat{\Delta}_{mk}(t-1) + \sqrt{\frac{2a\,\widehat{\sigma}^2_{mk}(t-1)}{T_{mk}(t-1)}} + \frac{7ab}{3\big(T_{mk}(t-1)-1\big)}.$$

Note that the exploration term in the B-index has now two components: the first one depends on the empirical variance and the second one decreases as $O(1/T_{mk})$. As a result, arms with low variance will be explored much less than in the GapE algorithm. Similar to the difference between UCB [3] and UCB-V [2], while the B-index in GapE is motivated by Hoeffding's inequalities, the one for GapE-V is obtained using an empirical Bernstein's inequality [11, 2]. The following performance bound can be proved for GapE-V algorithm. We report the proof of Theorem 2 in App. C of [8].

**Theorem 2.** *If GapE-V is run with parameter* $0 < a \leq \frac{8}{9}\frac{n-2MK}{H^\sigma}$, *then it satisfies*

$$\ell(n) \leq \mathbb{P}\big(\exists m : J_m(n) \neq k^*_m\big) \leq 6nMK \exp\left(-\frac{9a}{64 \times 64}\right)$$

*in particular for* $a = \frac{8}{9}\frac{n-2MK}{H^\sigma}$, *we have* $\ell(n) \leq 6nMK \exp\big(-\frac{1}{64\times 8}\frac{n-2MK}{H^\sigma}\big)$.

In Theorem 2, $H^\sigma$ is the complexity of the GapE-V algorithm and is defined as

$$H^\sigma = \sum_{m=1}^{M} \sum_{k=1}^{K} \frac{\big(\sigma_{mk} + \sqrt{\sigma^2_{mk} + (16/3)b\Delta_{mk}}\big)^2}{\Delta^2_{mk}}.$$

Although the variance-complexity $H^\sigma$ could be larger than the complexity $H$ used in GapE, whenever the variances of the arms are small compared to the range $b$ of the distribution, we expect $H^\sigma$ to be smaller than $H$. Furthermore, if the arms have very different variances, then GapE-V is expected to better capture the complexity of each arm and allocate the pulls accordingly. For instance, in the case where all the gaps are the same, GapE tends to allocate pulls proportionally to the complexity $H_{mk}$ and it would perform an almost uniform allocation over bandits and arms. On the other hand, the variances of the arms could be very heterogeneous and GapE-V would adapt the allocation strategy by pulling more often the arms whose values are more uncertain.

**Adaptive GapE and GapE-V.** A drawback of GapE and GapE-V is that the exploration parameter $a$ should be tuned according to the complexities $H$ and $H^\sigma$ of the multi-bandit problem, which are rarely known in advance. A straightforward solution to this issue is to move to an adaptive version of these algorithms by substituting $H$ and $H^\sigma$ with suitable estimates $\widehat{H}$ and $\widehat{H}^\sigma$. At each step $t$ of the adaptive GapE and GapE-V algorithms, we estimate these complexities as

$$\widehat{H}(t) = \sum_{m,k} \frac{b^2}{\text{UCB}_{\Delta_i}(t)^2}, \qquad \widehat{H}^\sigma(t) = \sum_{m,k} \frac{\big(\text{LCB}_{\sigma_i}(t) + \sqrt{\text{LCB}_{\sigma_i}(t)^2 + (16/3)b \times \text{UCB}_{\Delta_i}(t)}\big)^2}{\text{UCB}_{\Delta_i}(t)^2}, \quad \text{where}$$

$$\text{UCB}_{\Delta_i}(t) = \widehat{\Delta}_i(t-1) + \sqrt{\frac{1}{2T_i(t-1)}} \quad \text{and} \quad \text{LCB}_{\sigma_i}(t) = \max\left(0, \widehat{\sigma}_i(t-1) - \sqrt{\frac{2}{T_i(t-1)-1}}\right).$$

Similar to the adaptive version of UCB-E in [1], $\widehat{H}$ and $\widehat{H}^\sigma$ are lower-confidence bounds on the true complexities $H$ and $H^\sigma$. Note that the GapE and GapE-V bounds written for the optimal value of $a$ indicate an inverse relation between the complexity and the exploration. By using a lower-bound on the true $H$ and $H^\sigma$, the algorithms tend to explore arms more uniformly and this allows them to increase the accuracy of their estimated complexities. Although we do not analyze these algorithms, we empirically show in Sec. 4 that they are in fact able to match the performance of the GapE and GapE-V algorithms.

## 4 Numerical Simulations

In this section, we report numerical simulations of the gap-based algorithms presented in this paper, GapE and GapE-V, and their adaptive versions A-GapE and A-GapE-V, and compare them with *Unif*

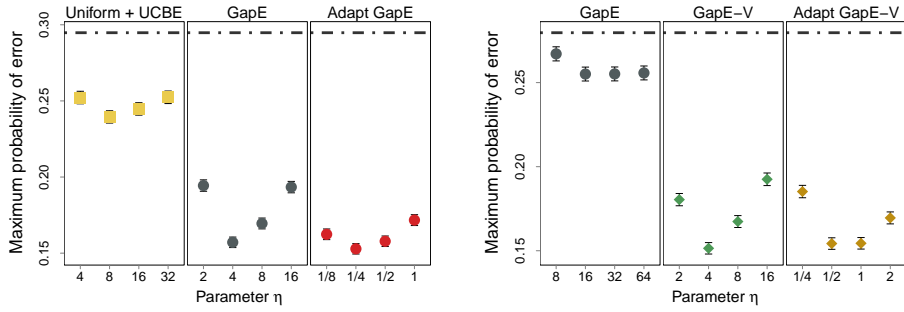

Figure 2: *(left)* Problem 1: Comparison between GapE, adaptive GapE, and the uniform strategies. *(right)* Problem 2: Comparison between GapE, GapE-V, and adaptive GapE-V algorithms.

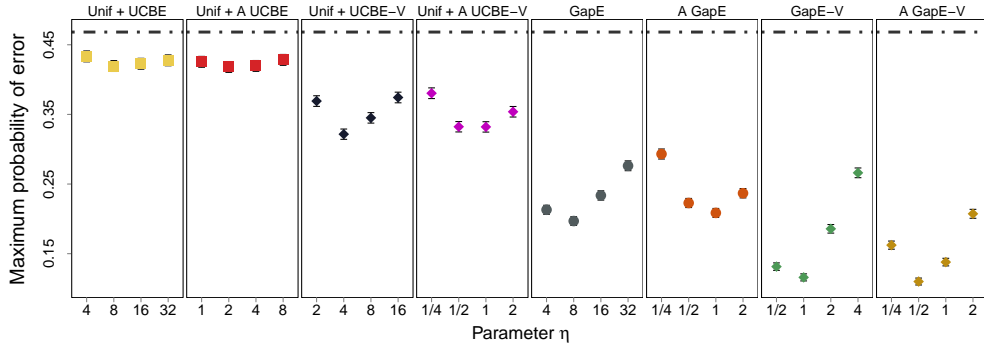

Figure 3: Performance of the algorithms in Problem 3.

and *Unif+UCB-E* algorithms introduced in Sec. 3.1. The results of our experiments both those in the paper and those in App. A of [8] indicate that **1)** GapE successfully adapts its allocation strategy to the complexity of each bandit and outperforms the uniform allocation strategies, **2)** the use of the empirical variance in GapE-V can significantly improve the performance over GapE, and **3)** the adaptive versions of GapE and GapE-V that estimate the complexities $H$ and $H^\sigma$ online attain the same performance as the basic algorithms, which receive $H$ and $H^\sigma$ as an input.

**Experimental setting.** We use the following three problems in our experiments. Note that $b = 1$ and that a Rademacher distribution with parameters $(x, y)$ takes value $x$ or $y$ with probability $1/2$.

• *Problem 1.* $n = 700$, $M = 2$, $K = 4$. The arms have Bernoulli distribution with parameters: *bandit 1* = $(0.5, 0.45, 0.4, 0.3)$, *bandit 2* = $(0.5, 0.3, 0.2, 0.1)$.
• *Problem 2.* $n = 1000$, $M = 2$, $K = 4$. The arms have Rademacher distribution with parameters $(x, y)$: *bandit 1* = $\{(0, 1.0), (0.45, 0.45), (0.25, 0.65), (0, 0.9)\}$ and in *bandit 2* = $\{(0.4, 0.6), (0.45, 0.45), (0.35, 0.55), (0.25, 0.65)\}$.
• *Problem 3.* $n = 1400$, $M = 4$, $K = 4$. The arms have Rademacher distribution with parameters $(x, y)$: *bandit 1* = $\{(0, 1.0), (0.45, 0.45), (0.25, 0.65), (0, 0.9)\}$, *bandit 2* = $\{(0.4, 0.6), (0.45, 0.45), (0.35, 0.55), (0.25, 0.65)\}$, *bandit 3* = $\{(0, 1.0), (0.45, 0.45), (0.25, 0.65), (0, 0.9)\}$, and *bandit 4* = $\{(0.4, 0.6), (0.45, 0.45), (0.35, 0.55), (0.25, 0.65)\}$.

All the algorithms, except the uniform allocation, have an exploration parameter $a$. The theoretical analysis suggests that $a$ should be proportional to $\frac{n}{H}$. Although $a$ could be optimized according to the bound, since the constants in the analysis are not accurate, we will run the algorithms with $a = \eta \frac{n}{H}$, where $\eta$ is a parameter which is empirically tuned (in the experiments we report four different values for $\eta$). If $H$ correctly defines the complexity of the exploration problem (i.e., the number of samples to find the best arms with high probability), $\eta$ should simply correct the inaccuracy of the constants in the analysis, and thus, the range of its nearly-optimal values should be constant across different problems. In *Unif+UCB-E*, UCB-E is run with the budget of $n/M$ and the same parameter $\eta$ for all the bandits. Finally, we set $n \simeq H^\sigma$, since we expect $H^\sigma$ to roughly capture the number of pulls necessary to solve the pure exploration problem with high probability. In Figs. 2 and 3, we report the performance $l(n)$, i.e. the probability to identify the best arm in all the bandits after $n$ rounds, of the gap-based algorithms as well as *Unif* and *Unif+UCB-E* strategies. The results are averaged

over $10^5$ runs and the error bars correspond to three times the estimated standard deviation. In all the figures the performance of *Unif* is reported as a horizontal dashed line.

The left panel of Fig. 2 displays the performance of *Unif+UCB-E*, GapE, and A-GapE in Problem 1. As expected, *Unif+UCB-E* has a better performance ($23.9\%$ probability of error) than *Unif* ($29.4\%$ probability of error), since it adapts the allocation within each bandit so as to pull more often the nearly-optimal arms. However, the two bandit problems are not equally difficult. In fact, their complexities are very different ($H_1 \simeq 925$ and $H_2 \simeq 67$), and thus, much less samples are needed to identify the best arm in the second bandit than in the first one. Unlike *Unif+UCB-E*, GapE adapts its allocation strategy to the complexities of the bandits (on average only $19\%$ of the pulls are allocated to the second bandit), and at the same time to the arm complexities within each bandit (in the first bandit the averaged allocation of GapE is $(37\%, 36\%, 20\%, 7\%)$). As a result, GapE has a probability of error of $15.7\%$, which represents a significant improvement over *Unif+UCB-E*.

The right panel of Fig. 2 compares the performance of GapE, GapE-V, and A-GapE-V in Problem 2. In this problem, all the gaps are equals ($\Delta_{mk} = 0.05$), thus all the arms (and bandits) have the same complexity $H_{mk} = 400$. As a result, GapE tends to implement a nearly uniform allocation, which results in a small difference between *Unif* and GapE ($28\%$ and $25\%$ accuracy, respectively). The reason why GapE is still able to improve over *Unif* may be explained by the difference between static and dynamic allocation strategies and it is further investigated in App. A of [8]. Unlike the gaps, the variance of the arms is extremely heterogeneous. In fact, the variance of the arms of bandit 1 is bigger than in bandit 2, thus making it harder to solve. This difference is captured by the definition of $H^\sigma$ ($H_1^\sigma \simeq 1400 > H_2^\sigma \simeq 600$). Note also that $H^\sigma \leq H$. As discussed in Sec. 3.2, since GapE-V takes into account the empirical variance of the arms, it is able to adapt to the complexity $H_{mk}^\sigma$ of each bandit-arm pair and to focus more on uncertain arms. GapE-V improves the final accuracy by almost $10\%$ w.r.t. GapE. From both panels of Fig. 2, we also notice that the adaptive algorithms achieve similar performance to their non-adaptive counterparts. Finally, we notice that a good choice of parameter $\eta$ for GapE-V is always close to 2 and 4 (see also [8] for additional experiments), while GapE needs $\eta$ to be tuned more carefully, particularly in Problem 2 where the large values of $\eta$ try to compensate the fact that $H$ does not successfully capture the real complexity of the problem. This further strengthens the intuition that $H^\sigma$ is a more accurate measure of the complexity for the multi-bandit pure exploration problem.

While Problems 1 and 2 are relatively simple, we report the results of the more complicated Problem 3 in Fig. 3. The experiment is designed so that the complexity w.r.t. the variance of each bandit and within each bandit is strongly heterogeneous. In this experiment, we also introduce UCBE-V that extends UCB-E by taking into account the empirical variance similarly to GapE-V. The results confirm the previous findings and show the improvement achieved by introducing empirical estimates of the variance and allocating non-uniformly over bandits.

## 5  Conclusion

In this paper, we studied the problem of best arm identification in a multi-bandit multi-armed setting. We introduced a gap-based exploration algorithm, called GapE, and proved an upper-bound for its probability of error. We extended the basic algorithm to also consider the variance of the arms and proved an upper-bound for its probability of error. We also introduced adaptive versions of these algorithms that estimate the complexity of the problem online. The numerical simulations confirmed the theoretical findings that GapE and GapE-V outperform other allocation strategies, and that their adaptive counterparts are able to estimate the complexity without worsening the global performance.

Although GapE does not know the gaps, the experimental results reported in [8] indicate that it might outperform a static allocation strategy, which knows the gaps in advance, thus suggesting that an adaptive strategy could perform better than a static one. This observation asks for further investigation. Moreover, we plan to apply the algorithms introduced in this paper to the problem of rollout allocation for classification-based policy iteration in reinforcement learning [9, 6], where the goal is to identify the greedy action (*arm*) in each of the states (*bandit*) in a training set.

**Acknowledgments** Experiments presented in this paper were carried out using the Grid'5000 experimental testbed (https://www.grid5000.fr). This work was supported by Ministry of Higher Education and Research, Nord-Pas de Calais Regional Council and FEDER through the "contrat de projets état region 2007–2013", French National Research Agency (ANR) under project LAMPADA $n°$ ANR-09-EMER-007, European Community's Seventh Framework Programme (FP7/2007-2013) under grant agreement $n°$ 231495, and PASCAL2 European Network of Excellence.

# References

[1] J.-Y. Audibert, S. Bubeck, and R. Munos. Best arm identification in multi-armed bandits. In *Proceedings of the Twenty-Third Annual Conference on Learning Theory*, pages 41–53, 2010.

[2] Jean-Yves Audibert, Rémi Munos, and Csaba Szepesvári. Tuning bandit algorithms in stochastic environments. In Marcus Hutter, Rocco Servedio, and Eiji Takimoto, editors, *Algorithmic Learning Theory*, volume 4754 of *Lecture Notes in Computer Science*, pages 150–165. Springer Berlin / Heidelberg, 2007.

[3] P. Auer, N. Cesa-Bianchi, and P. Fischer. Finite-time analysis of the multi-armed bandit problem. *Machine Learning*, 47:235–256, 2002.

[4] S. Bubeck, R. Munos, and G. Stoltz. Pure exploration in multi-armed bandit problems. In *Proceedings of the Twentieth International Conference on Algorithmic Learning Theory*, pages 23–37, 2009.

[5] K. Deng, J. Pineau, and S. Murphy. Active learning for personalizing treatment. In *IEEE Symposium on Adaptive Dynamic Programming and Reinforcement Learning*, 2011.

[6] C. Dimitrakakis and M. Lagoudakis. Rollout sampling approximate policy iteration. *Machine Learning Journal*, 72(3):157–171, 2008.

[7] Eyal Even-Dar, Shie Mannor, and Yishay Mansour. Action elimination and stopping conditions for the multi-armed bandit and reinforcement learning problems. *Journal of Machine Learning Research*, 7:1079–1105, 2006.

[8] V. Gabillon, M. Ghavamzadeh, A. Lazaric, and S. Bubeck. Multi-bandit best arm identification. Technical Report 00632523, INRIA, 2011.

[9] M. Lagoudakis and R. Parr. Reinforcement learning as classification: Leveraging modern classifiers. In *Proceedings of the Twentieth International Conference on Machine Learning*, pages 424–431, 2003.

[10] O. Maron and A. Moore. Hoeffding races: Accelerating model selection search for classification and function approximation. In *Proceedings of Advances in Neural Information Processing Systems 6*, 1993.

[11] A. Maurer and M. Pontil. Empirical bernstein bounds and sample-variance penalization. In *22th annual conference on learning theory*, 2009.

[12] V. Mnih, Cs. Szepesvári, and J.-Y. Audibert. Empirical Bernstein stopping. In *Proceedings of the Twenty-Fifth International Conference on Machine Learning*, pages 672–679, 2008.

[13] H. Robbins. Some aspects of the sequential design of experiments. *Bulletin of the American Mathematics Society*, 58:527–535, 1952.

